# Dependence of Orientation Tuning on Recurrent Excitation and Inhibition in a Network Model of V1

**Klaus Wimmer**[1]*, **Marcel Stimberg**[1]*, **Robert Martin**[1], **Lars Schwabe**[2], **Jorge Mariño**[3], **James Schummers**[4], **David C. Lyon**[5], **and Klaus Obermayer**[1]

[1] Bernstein Center for Computational Neuroscience and Technische Universität Berlin, Germany
[2] Dept of Computer Science and Electrical Engineering, University of Rostock, Germany
[3] Dept of Medicine, Neuroscience, and Motor Control Group, Univ. A Coruña, Spain
[4] Dept of Brain and Cognitive Sci and Picower Ctr for Learning and Memory, MIT, Cambridge
[5] Dept of Anatomy and Neurobiology, University of California, Irvine, USA
`[klaus, mst]@cs.tu-berlin.de`

## Abstract

The computational role of the local recurrent network in primary visual cortex is still a matter of debate. To address this issue, we analyze intracellular recording data of cat V1, which combine measuring the tuning of a range of neuronal properties with a precise localization of the recording sites in the orientation preference map. For the analysis, we consider a network model of Hodgkin-Huxley type neurons arranged according to a biologically plausible two-dimensional topographic orientation preference map. We then systematically vary the strength of the recurrent excitation and inhibition relative to the strength of the afferent input. Each parametrization gives rise to a different model instance for which the tuning of model neurons at different locations of the orientation map is compared to the experimentally measured orientation tuning of membrane potential, spike output, excitatory, and inhibitory conductances. A quantitative analysis shows that the data provides strong evidence for a network model in which the afferent input is dominated by strong, balanced contributions of recurrent excitation and inhibition. This recurrent regime is close to a regime of "instability", where strong, self-sustained activity of the network occurs. The firing rate of neurons in the best-fitting network is particularly sensitive to small modulations of model parameters, which could be one of the functional benefits of a network operating in this particular regime.

## 1 Introduction

One of the major tasks of primary visual cortex (V1) is the computation of a representation of orientation in the visual field. Early models [1], combining the center-surround receptive fields of lateral geniculate nucleus to give rise to orientation selectivity, have been shown to be over-simplistic [2; 3]. Nonetheless, a debate remains regarding the contribution of afferent and recurrent excitatory and inhibitory influences [4; 5]. Information processing in cortex changes dramatically with this "cortical operating regime", i. e. depending on the relative strengths of the afferent and the different recurrent inputs [6; 7]. Recently, experimental and theoretical studies have investigated how a cell's orientation tuning depends on its position in the orientation preference map [7–10]. However, the computation of orientation selectivity in primary visual cortex is still a matter of debate.

The wide range of models operating in different regimes that are discussed in the literature are an indication that models of V1 orientation selectivity are underconstrained. Here, we assess whether the specific location dependence of the tuning of internal neuronal properties can provide sufficient

constraints to determine the corresponding cortical operating regime. The data originates from intracellular recordings of cat V1 [9], combined with optical imaging. This allowed to measure, *in vivo*, the output (firing rate) of neurons, the input (excitatory and inhibitory conductances) and a state variable (membrane potential) as a function of the position in the orientation map. Figure 1 shows the experimentally observed tuning strength of each of these properties depending on the distribution of orientation selective cells in the neighborhood of each neuron. The x-axis spans the range from pinwheels (0) to iso-orientation domains (1), and each y-axis quantifies the sharpness of tuning of the individual properties (see section 2.2). The tuning of the membrane potential ($V_m$) as well as the tuning of the total excitatory ($g_e$) and inhibitory ($g_i$) conductances vary strongly with map location, whereas the tuning of the firing rate ($f$) does not. Specifically, the conductances and the membrane potential are sharper tuned for neurons within an iso-orientation domain, where the neighboring neurons have very similar orientation preferences, as compared to neurons close to a pinwheel center, where the neighboring neurons show a broad range of orientation preferences.

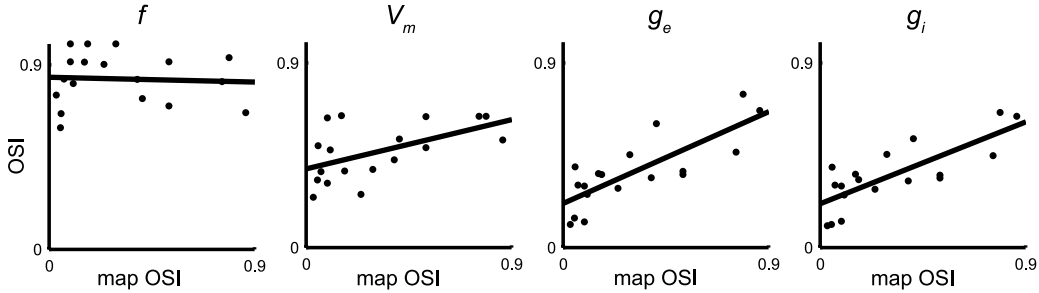

Figure 1: Variation of the orientation selectivity indices (OSI, cf. Equation 2) of the firing rate ($f$), the average membrane potential ($V_m$), and the excitatory ($g_e$) and inhibitory ($g_i$) input conductances of neurons in cat V1 with the map OSI (the orientation selectivity index of the orientation map at the location of the measured neuron). Dots indicate the experimentally measured values from 18 cells [9]. Solid lines show the result of a linear regression. The slopes (values $\pm$ 95% confidence interval) are $-0.02 \pm 0.24$ ($f$), $0.27 \pm 0.22$ ($V_m$), $0.49 \pm 0.20$ ($g_e$), $0.44 \pm 0.19$ ($g_i$).

This paper focuses on the constraints that this specific map-location dependence of neuronal properties imposes on the operating regime of a generic network composed of Hodgkin-Huxley type model neurons. The model takes into account that the lateral inputs a cell receives are determined (1) by the position in the orientation map and (2) by the way that synaptic inputs are pooled across the map. The synaptic pooling radius has been shown experimentally to be independent of map location [9], resulting in essentially different local recurrent networks depending on whether the neighborhood is made up of neurons with similar preferred orientation, such as in an iso-orientation domain, or is highly non-uniform, such as close to a pinwheel. The strength of lateral connections, on the other hand, is unknown. Mariño et al. [9] have shown that their data is compatible with a model showing strong recurrent excitation and inhibition. However, this approach cannot rule out alternative explanations accounting for the emergence of orientation tuning in V1. Here, we systematically explore the model space, varying the strength of the recurrent excitation and inhibition. This, in effect, allows us to test the full range of models, including feed-forward-, inhibition- and excitation-dominated models as well as balanced recurrent models, and to determine those that are compatible with the observed data.

## 2 Methods

### 2.1 Simulation: The Hodgkin-Huxley network model

The network consists of Hodgkin-Huxley type point neurons and includes three voltage dependent currents (Na$^+$ and K$^+$ for generation of action potentials, and a non-inactivating K$^+$-current that is responsible for spike-frequency adaptation). Spike-frequency adaptation was reduced by a factor 0.1 for inhibitory neurons. For a detailed description of the model neuron and the parameter values, see Destexhe et al. [11]. Every neuron receives afferent, recurrent and background input. We

used exponential models for the synaptic conductances originating from GABA$_\text{A}$-like inhibitory and AMPA-like excitatory synapses [12]. Slow NMDA-like excitatory synapses are modeled by a difference of two exponentials (parameters are summarized in Table 1). Additional conductances represent background activity (Ornstein-Uhlenbeck conductance noise, cf. Destexhe et al. [11]).

Table 1: Parameters of the Hodgkin-Huxley type neural network.

| PARAMETER | DESCRIPTION | VALUE |
|---|---|---|
| $N_\text{Aff}$ | Number of afferent exc. synaptic connections per cell | 20 |
| $N_E$ | Number of recurrent exc. synaptic connections per cell | 100 |
| $N_I$ | Number of recurrent inh. synaptic connections per cell | 50 |
| $\sigma_E = \sigma_I$ | Spread of recurrent connections (std. dev.) | 4 units (125 μm) |
| $E_e$ | Reversal potential excitatory synapses | 0 mV |
| $E_i$ | Reversal potential inhibitory synapses | -80 mV |
| $\tau_E$ | Time constant of AMPA-like synapses | 5 ms |
| $\tau_I$ | Time constant of GABA$_\text{A}$-like synapses | 5 ms |
| $\tau_1$ | Time constant of NMDA-like synapses | 80 ms |
| $\tau_2$ | Time constant of NMDA-like synapses | 2 ms |
| $\mu_E^d, \sigma_E^d$ | Mean and standard deviation of excitatory synaptic delay | 4 ms, 2 ms |
| $\mu_I^d, \sigma_I^d$ | Mean and standard deviation of inhibitory synaptic delay | 1.25 ms, 1 ms |
| $\overline{g}_\text{E}^\text{Aff}$ | Peak conductance of afferent input to exc. cells | 141 nS |
| $\overline{g}_\text{I}^\text{Aff}$ | Peak conductance of afferent input to inh. cells | 0.73 $\overline{g}_\text{E}^\text{Aff}$ |
| $\overline{g}_\text{II}$ | Peak conductance from inh. to inh. cells | 1.33 $\overline{g}_\text{E}^\text{Aff}$ |
| $\overline{g}_\text{EI}$ | Peak conductance from inh. to exc. cells | 1.33 $\overline{g}_\text{E}^\text{Aff}$ |

The network was composed of 2500 excitatory cells arranged on a $50 \times 50$ grid and 833 inhibitory neurons placed at random grid locations, thus containing 75% excitatory and 25% inhibitory cells. The complete network modeled a patch of cortex $1.56 \times 1.56 \, \text{mm}^2$ in size. Connection probabilities for all recurrent connections (between the excitatory and inhibitory population and within the populations) were determined from a spatially isotropic Gaussian probability distribution (for parameters, see Table 1) with the same spatial extent for excitation and inhibition, consistent with experimental measurements [9]. In order to avoid boundary effects, we used periodic boundary conditions. Recurrent excitatory conductances were modeled as arising from 70% fast (AMPA-like) versus 30% slow (NMDA-like) receptors. If a presynaptic neuron generated a spike, this spike was transferred to the postsynaptic neuron with a certain delay (parameters are summarized in Table 1).

The afferent inputs to excitatory and inhibitory cortical cells were modeled as Poisson spike trains with a time-independent firing rate $f_\text{Aff}$ given by

$$f_\text{Aff}(\theta_\text{stim}) = 30 \, \text{Hz} \left( r_\text{base} + (1 - r_\text{base}) \exp \left( -\frac{(\theta_\text{stim} - \theta)^2}{(2\sigma_\text{Aff})^2} \right) \right), \tag{1}$$

where $\theta_\text{stim}$ is the orientation of the presented stimulus, $\theta$ is the preferred orientation of the cell, $r_\text{base} = 0.1$ is a baseline firing rate, and $\sigma_\text{Aff} = 27.5°$ is the tuning width. These input spike trains exclusively trigger fast, AMPA-like excitatory synapses. The orientation preference for each neuron was assigned according to its location in an artificial orientation map (Figure 2A). This map was calibrated such that the pinwheel distance and the spread of recurrent connections matches experimental data [9].

In order to measure the orientation tuning curves of $f$, $V_m$, $g_e$, and $g_i$, the response of the network to inputs with different orientations was computed for $1.5 \, \text{s}$ with $0.25 \, \text{ms}$ resolution (usually, the network settled into a steady state after a few hundred milliseconds). We then calculated the firing rate, the average membrane potential, and the average total excitatory and inhibitory conductances for every cell in an interval between $0.5 \, \text{s}$ and $1.5 \, \text{s}$.

## 2.2 Quantitative evaluation: Orientation selectivity index (OSI) and OSI-OSI slopes

We analyze orientation tuning using the orientation selectivity index [13], which is given by

$$\text{OSI} = \sqrt{\left( \sum_{i=1}^N R(\phi_i) \cos(2\phi_i) \right)^2 + \left( \sum_{i=1}^N R(\phi_i) \sin(2\phi_i) \right)^2} \Big/ \sum_{i=1}^N R(\phi_i). \tag{2}$$

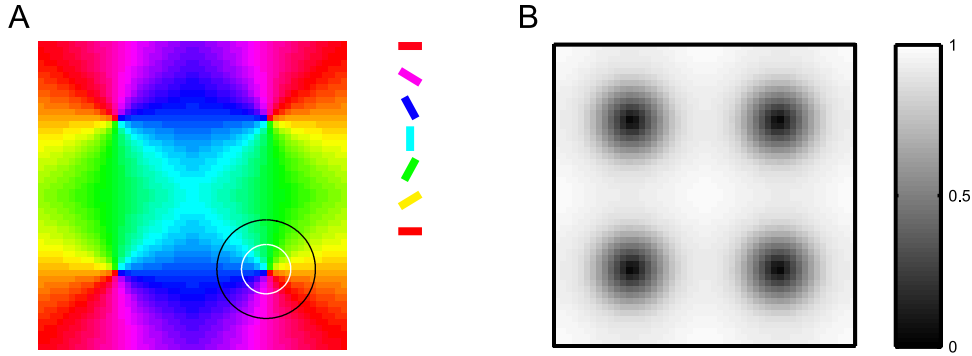

Figure 2: (A) Artificial orientation map with four pinwheels of alternating handedness arranged on a 2-dimensional grid. The white (black) circle denotes the one-(two-) $\sigma$-area corresponding to the radial Gaussian synaptic connection profile ($\sigma_E = \sigma_I = 125\,\mu m$). (B) Map OSI of the artificial orientation map. Pinwheel centers appear in black.

$R(\phi_i)$ is the value of the quantity whose tuning is considered, in response to a stimulus of orientation $\phi_i$ (e.g. the spiking activity). For all measurements, eight stimulus orientations $\phi_i \in \{-67.5, -45, -22.5, 0, 22.5, 45, 67.5, 90\}$ were presented. The OSI is then a measure of tuning sharpness ranging from 0 (unselective) to 1 (perfectly selective). In addition, the OSI was used to characterize the sharpness of the recurrent input a cell receives based on the orientation preference map. To calculate this *map OSI*, we estimate the local orientation preference distribution by binning the orientation preference of all pixels within a radius of $250\,\mu m$ around a cell into bins of $10°$ size; the number of cells in each bin replaces $R(\phi_i)$. Figure 2 shows the artificial orientation map and the map OSI for the cells in our network model. The map OSI ranges from almost 0 for cells close to pinwheel centers to almost 1 in the linear zones of the iso-orientation domains.

The dependence of each tuning property on the local map OSI was then described by a linear regression line using the least squares method. These linear fits provided a good description of the relationship between map OSI and the tuning of the neuronal properties in the simulations (mean squared deviation around the regression lines was typically below 0.0025 and never above a value of 0.015) as well as in the experimental data (mean squared deviation was between 0.009 ($g_i$) and 0.015 ($f$)). In order to find the regions of parameter space where the linear relationship predicted by the models is compatible with the data, the confidence interval for the slope of the linear fit to the data was used.

## 3 Results

The parameter space of the class of network models considered in this paper is spanned by the peak conductance of synaptic excitatory connections to excitatory ($\bar{g}_{\mathrm{EE}}$) and inhibitory ($\bar{g}_{\mathrm{IE}}$) neurons. We shall first characterize the operating regimes found in this model space, before comparing the location dependence of tuning observed in the different models with that found experimentally.

### 3.1 Operating regimes of the network model

The operating regimes of a firing rate model can be defined in terms of the strength and shape of the effective recurrent input [7]. The definitions of Kang et al. [7], however, are based on the analytical solution of a linear firing rate model where all neurons are above threshold and cannot be applied to the non-linear Hodgkin-Huxley network model used here. Therefore, we characterize the parameter space explored here using a numerical definition of the operating regimes. This definition is based on the orientation tuning of the input currents to the excitatory model cells in the orientation domain ($0.6 <$ map OSI $< 0.9$). Specifically, if the sum of input currents is positive (negative) for all presented orientations, recurrent excitation (inhibition) is dominant, and the regime thus excitatory (EXC; respective inhibitory, INH). If the sum of input currents has a positive maximum and a negative minimum (i.e. Mexican-hat like), a model receives significant excitation as well as inhibi-

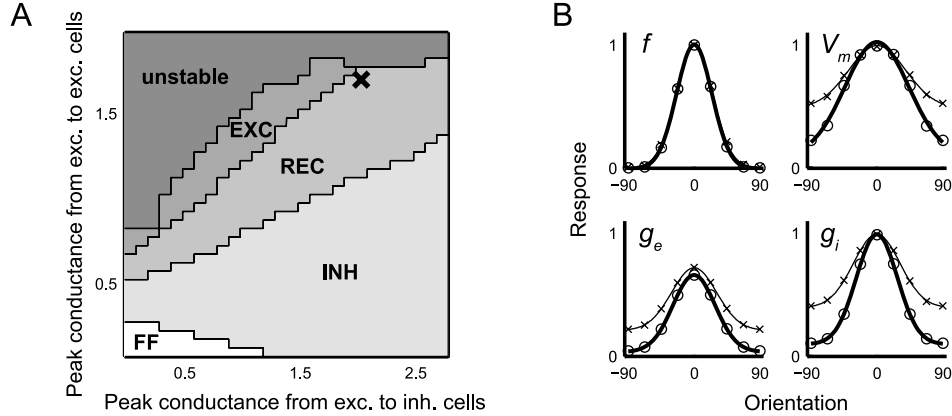

Figure 3: (A) Operating regimes of the network model as a function of the peak conductance of synaptic excitatory connections to excitatory ($\overline{g}_{\mathrm{EE}}$) and inhibitory ($\overline{g}_{\mathrm{IE}}$) neurons: FF – feed-forward, EXC – recurrent excitatory dominated, INH – recurrent inhibitory dominated, REC – strong recurrent excitation and inhibition, and unstable. The conductances are given as multiples of the afferent peak conductance of excitatory neurons ($\overline{g}_{\mathrm{E}}^{\mathrm{Aff}}$). The figure summarizes simulation results for $38 \times 28$ different values of $\overline{g}_{\mathrm{EE}}$ and $\overline{g}_{\mathrm{IE}}$. (B) Tuning curves for one example network in the REC regime (marked by a cross in A). Mean responses across cells are shown for the firing rate ($f$), the membrane potential ($V_m$), the total excitatory ($g_e$), and the total inhibitory conductance ($g_i$), separately for cells in iso-orientation domains ($0.6 <$ map OSI $< 0.9$, thick lines) and cells close to pinwheel centers (map OSI $< 0.3$, thin lines). For each cell, responses were individually aligned to its preferred orientation and normalized to its maximum response; for the $V_m$ tuning curve, the mean membrane potential without any stimulation ($V_m = -64.5\,\mathrm{mV}$) was subtracted beforehand. To allow comparison of the magnitude of $g_i$ and $g_e$ responses, both types of conductances were normalized to the maximum $g_i$ response.

tion and we refer to such a model as operating in the recurrent regime (REC). An example for the orientation tuning properties observed in the recurrent regime is shown in Figure 3B. Finally, if the sum of the absolute values of the currents through excitatory and inhibitory recurrent synapses of the model cells (at preferred orientation) is less than 30% of the current through afferent synapses, the afferent drive is dominant and we call such regimes feed-forward (FF).

The regions of parameter space corresponding to these operating regimes are depicted in Figure 3A as a function of the peak conductance of synaptic excitatory connections to excitatory ($\overline{g}_{\mathrm{EE}}$) and inhibitory ($\overline{g}_{\mathrm{IE}}$) neurons. We refer to the network as "unstable" if the model neurons show strong responses (average firing rate exceeds $100\,\mathrm{Hz}$) and remain at high firing rates if the afferent input is turned off; i. e. the network shows self-sustained activity. In this regime, the model neurons lose their orientation tuning.

### 3.2 Orientation tuning properties in the different operating regimes

We analyzed the dependence of the orientation tuning properties on the operating regimes and compared them to the experimental data. For every combination of $\overline{g}_{\mathrm{EE}}$ and $\overline{g}_{\mathrm{IE}}$, we simulated the responses of neurons in the network model to oriented stimuli in order to measure the orientation tuning of $V_m$, $f$, $g_e$ and $g_i$ (see Methods). The OSI of each of the four quantities can then be plotted against the map OSI to reveal the dependence of the tuning on the map location (similar to the experimental data shown in Figure 1). The slope of the linear regression of this OSI-OSI dependence was used to characterize the different operating points of the network. Figure 4 shows these slopes for the tuning of $f$, $V_m$, $g_e$ and $g_i$, as a function of $\overline{g}_{\mathrm{EE}}$ and $\overline{g}_{\mathrm{IE}}$ of the respective Hodgkin-Huxley network models (gray scale). Model networks with strong recurrent excitation (large values of $\overline{g}_{\mathrm{EE}}$), as in the REC regime, predict steeper slopes than networks with less recurrent excitation. In other words, as the regime becomes increasingly more recurrently dominated, the recurrent contribution leads to sharper tuning in neurons within iso-orientation domains as compared to neurons near the

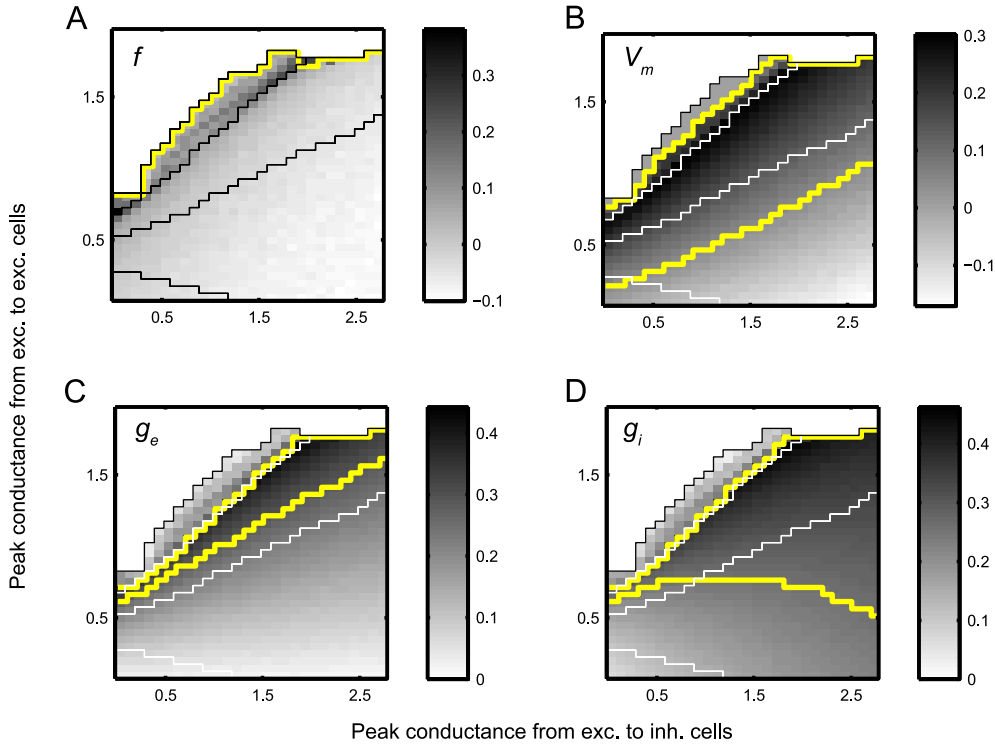

Figure 4: Location dependence of orientation tuning of the conductances, the membrane potential, and the firing rate in the network model. The figure shows the slope values of the OSI-OSI regression lines (in gray values) as a function of the peak conductance of synaptic excitatory connections to excitatory ($\overline{g}_{EE}$) and inhibitory ($\overline{g}_{IE}$) neurons, separately for the spike rate (A), the membrane potential (B), the total synaptic excitatory (C), and inhibitory conductance (D). The conductances are given as multiples of the afferent peak conductance of excitatory neurons ($\overline{g}_E^{Aff}$). Thin lines denote the borders of the different operating regimes (cf. Figure 3). The region delimited by the thick yellow line corresponds to slope values within the 95% confidence interval of the corresponding experimental data. Note that in (A) this region covers the whole range of operating regimes except the unstable regime.

pinwheel centers. However, yet closer to the line of instability the map-dependence of the tuning almost vanishes (slope approaching zero). This reflects the strong excitatory recurrent input in the EXC regime which leads to an overall increase in the network activity that is almost untuned and therefore provides very similar input to all neurons, regardless of map location. Also, the strongly inhibitory-dominated regimes (large values of $\overline{g}_{IE}$) at the bottom right corner of Figure 4 are of interest. Here, the slope of the location dependence becomes negative for the tuning of firing rate $f$ and membrane potential $V_m$. Such a sharpening of the tuning close to pinwheels in an inhibition dominated regime has been observed elsewhere [8].

Comparing the slope of the OSI-OSI regression lines to the 95% confidence interval of the slopes estimated from the experimental data (Figure 1) allows us to determine those regions in parameter space that are compatible with the data (yellow contours in Figure 4). The observed location-independence of the firing rate tuning is compatible with all stable models in the parameter space (Figure 4A) and therefore does not constrain the model class. In contrast to this, the observed location-dependence of the membrane potential tuning (Figure 4B) and the inhibitory conductance tuning (Figure 4D) excludes most of the feed-forward and about half of the inhibitory-dominated regime. Most information, however, is gained from the observed location-dependence of the excitatory conductance tuning (Figure 4C). It constrains the network to operate in either a recurrent regime with strong excitation and inhibition or in a slightly excitatory-dominated regime.

### 3.3 Only the strongly recurrent regime satisfies all constraints

Combining the constraints imposed by the OSI-OSI relationship of the four measured quantities (yellow contour in both panels of Figure 5), we can conclude that the experimental data constrains the network to operate in a recurrent operating regime, with recurrent excitation and inhibition strong, approximately balanced, and dominating the afferent input. In addition, we calculated the sum of squared differences between the data points (Figure 1) and the OSI-OSI relationship predicted by the model, for each operating regime. The "best fitting" operating regime, which had the lowest squared difference, is marked with a cross in Figure 5. The corresponding simulated tuning curves for orientation domain and pinwheel cells are shown in Figure 3B.

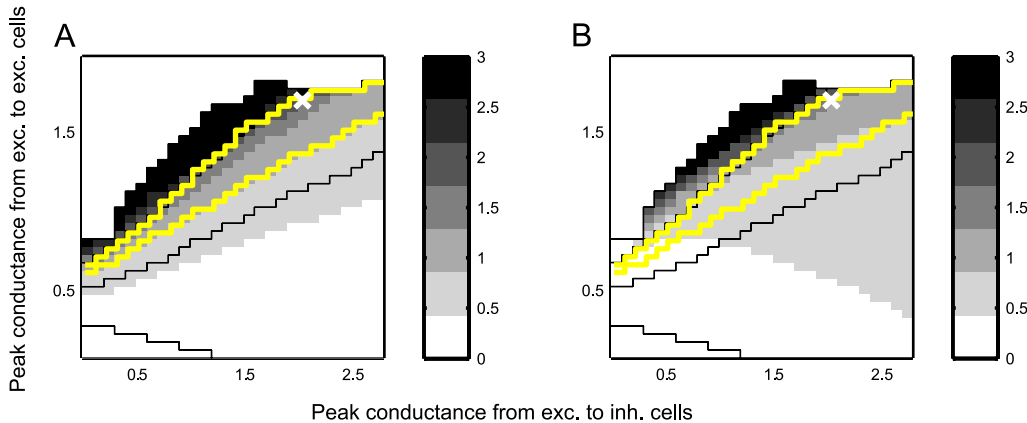

Figure 5: Ratio between (A) the excitatory current through the recurrent synapses and the current through afferent synapses of excitatory model cells and between (B) the inhibitory recurrent and the excitatory afferent current (in gray values). Currents were calculated for stimuli at the cells' preferred orientations, and averaged over all model cells within orientation domains $(0.6 < \text{map OSI} < 0.9)$. The region delimited by the thick yellow line corresponds to slope values that are in the 95% confidence interval for each experimentally measured quantity (spike rate, membrane potential, the total synaptic excitatory, and inhibitory conductance). The white cross at $(2.0, 1.7)$ denotes the combination of model parameters that yields the best fit to the experimental data (see text). Thin lines denote the borders of the different operating regimes (cf. Figure 3).

In line with the definition of the operating regimes, the excitatory current through the recurrent synapses (gray values in Figure 5A) plays a negligible role in the feed-forward and in most of the inhibitory-dominated regimes. Only in the recurrent and the excitatory-dominated regime is the recurrent current stronger than the afferent current. A similar observation holds for the inhibitory current (Figure 5B). The strong recurrent currents in the excitatory-dominated regime reflect the strong overall activity that reduce the map-location dependence of the total excitatory and inhibitory conductances (cf. Figure 4C and D).

## 4  Discussion

Although much is known about the anatomy of lateral connections in the primary visual cortex of cat, the *strengths* of synapses formed by short-range connections are largely unknown. In our study, we use intracellular physiological measurements to constrain the strengths of these connections. Extensively exploring the parameter space of a spiking neural network model, we find that neither feed-forward dominated, nor recurrent excitatory- or inhibitory-dominated networks are consistent with the tuning properties observed *in vivo*. We therefore conclude that the cortical network in cat V1 operates in a regime with a dominant recurrent influence that is approximately balanced between inhibition and excitation.

The analysis presented here focuses on the steady state the network reaches when presented with one non-changing orientation. In this light, it is very interesting, that a comparable operating regime has been indicated in an analysis of the dynamic properties of orientation tuning in cat V1 [14].

Our main finding – tuning properties of cat V1 are best explained by a network operating in a regime with strong recurrent excitation and inhibition – is robust against variation of the values chosen for other parameters not varied here, e. g. $\overline{g}_{\text{II}}$ and $\overline{g}_{\text{EI}}$ (data not shown). Nevertheless, the network architecture is based on a range of basic assumptions: e. g. all neurons in the network receive equally sharply tuned input. The explicit inclusion of location dependence of the input tuning might well lead to tuning properties compatible with the experimental data in different operating regimes. However, there is no evidence supporting such a location dependence of the afferent input and therefore assuming location-independent input seemed the most prudent basis for this analysis. Another assumption is the absence of untuned inhibition, since the inhibitory neurons in the network presented here receive tuned afferent input, too. The existence of an untuned inhibitory subpopulation is still a matter of debate (compare e. g. [15] and [16]). Naturally, such an untuned component would considerably reduce the location dependence of the inhibitory conductance $g_i$. Given that in our exploration only a small region of parameter space exists where the slope of $g_i$ is steeper than in the experiment, a major contribution of such an untuned inhibition seems incompatible with the data.

Our analysis demonstrates that the network model is compatible with the data only if it operates in a regime that – due to the strong recurrent connections – is close to instability. Such a network is very sensitive to changes in its governing parameters, e. g. small changes in connection strengths lead to large changes in the overall firing rate: In the regimes close to the line of instability, increasing $\overline{g}_{\text{EE}}$ by just 5% typically leads to increases in firing rate of around 40% (EXC), respectively 20% (REC). In the other regimes (FF and INH) firing rate only changes by around 2–3%. In the "best fitting" operating regime, a 10% change in firing rate, which is of similar magnitude as observed firing rate changes under attention in macaque V1 [17], is easily achieved by increasing $\overline{g}_{\text{EE}}$ by 2%. It therefore seems plausible that one benefit of being in such a regime is the possibility of significantly changing the "operating point" of the network through only small adjustments of the underlying parameters. Candidates for such an adjustment could be contextual modulations, adaptation or attentional effects.

The analysis presented here is based on data for cat V1. However, the ubiquitous nature of some of the architectural principles in neocortex suggests that our results may generalize to other cortical areas, functions and species.

## Footnotes

*K. Wimmer and M. Stimberg contributed equally to this work.

## References

[1] Hubel, D. H & Wiesel, T. N. (1962) *J Physiol* **160**, 106–154.

[2] Sompolinsky, H & Shapley, R. (1997) *Curr Opin Neurobiol* **7**, 514–522.

[3] Ferster, D & Miller, K. D. (2000) *Annu Rev Neurosci* **23**, 441–471.

[4] Martin, K. A. C. (2002) *Curr Opin Neurobiol* **12**, 418–425.

[5] Teich, A. F & Qian, N. (2006) *J Neurophysiol* **96**, 404–419.

[6] Ben-Yishai, R, Bar-Or, R. L, & Sompolinsky, H. (1995) *Proc Natl Acad Sci U S A* **92**, 3844–3848.

[7] Kang, K, Shelley, M, & Sompolinsky, H. (2003) *Proc Natl Acad Sci U S A* **100**, 2848–2853.

[8] McLaughlin, D, Shapley, R, Shelley, M, & Wielaard, D. J. (2000) *Proc Natl Acad Sci U S A* **97**, 8087–92.

[9] Mariño, J, Schummers, J, Lyon, D. C, Schwabe, L, Beck, O, Wiesing, P, Obermayer, K, & Sur, M. (2005) *Nat Neurosci* **8**, 194–201.

[10] Nauhaus, I, Benucci, A, Carandini, M, & Ringach, D. L. (2008) *Neuron* **57**, 673–679.

[11] Destexhe, A, Rudolph, M, Fellous, J, & Sejnowski, T. (2001) *Neuroscience* **107**, 13–24.

[12] Destexhe, A, Mainen, Z. F, & Sejnowski, T. J. (1998) in *Methods in neuronal modeling*, eds. Koch, C & Segev, I. (MIT Press, Cambridge, Mass), 2nd edition, pp. 1–25.

[13] Swindale, N. V. (1998) *Biol Cybern* **78**, 45–56.

[14] Schummers, J, Cronin, B, Wimmer, K, Stimberg, M, Martin, R, Obermayer, K, Koerding, K, & Sur, M. (2007) *Frontiers in Neuroscience* **1**, 145–159.

[15] Cardin, J. A, Palmer, L. A, & Contreras, D. (2007) *J Neurosci* **27**, 10333–10344.

[16] Nowak, L. G, Sanchez-Vives, M. V, & McCormick, D. A. (2008) *Cereb Cortex* **18**, 1058–1078.

[17] McAdams, C. J & Maunsell, J. H. (1999) *J Neurosci* **19**, 431–441.

